# Learning in higher-order 'artificial dendritic trees'

**Tony Bell**
Artificial Intelligence Laboratory
Vrije Universiteit Brussel
Pleinlaan 2, B-1050 Brussels, BELGIUM
(tony@arti.vub.ac.be)

## ABSTRACT

If neurons sum up their inputs in a non-linear way, as some simulations suggest, how is this distributed fine-grained non-linearity exploited during learning? How are all the small sigmoids in synapse, spine and dendritic tree lined up in the right areas of their respective input spaces? In this report, I show how an abstract atemporal highly nested tree structure with a quadratic transfer function associated with each branchpoint, can self organise using only a single global reinforcement scalar, to perform binary classification tasks. The procedure works well, solving the 6-multiplexer and a difficult phoneme classification task as well as back-propagation does, and faster. Furthermore, it does not calculate an error gradient, but uses a statistical scheme to build moving models of the reinforcement signal.

## 1. INTRODUCTION

The computational territory between the linearly summing McCulloch-Pitts neuron and the non-linear differential equations of Hodgkin & Huxley is relatively sparsely populated. Connectionists use variants of the former and computational neuroscientists struggle with the exploding parameter spaces provided by the latter. However, evidence from biophysical simulations suggests that the voltage transfer properties of synapses, spines and dendritic membranes involve many detailed non-linear interactions, not just a squashing function at the cell body. Real neurons may indeed be higher-order nets.

For the computationally-minded, higher order interactions means, first of all, quadratic terms. This contribution presents a simple learning principle for a binary tree with a logistic/quadratic transfer function at each node. These functions, though highly nested, are shown to be capable of changing their shape in concert. The resulting tree structure receives inputs at its leaves, and outputs an estimate of the probability that the input pattern is a member of one of two classes at the top.

A number of other schemes exist for learning in higher-order neural nets. Sigma-Pi units, higher-order threshold logic units (Giles & Maxwell, 87) and product units (Durbin & Rumelhart, 89) are all examples of units which learn coefficients of non-linear functions. Product unit networks, like Radial Basis Function nets, consist of a layer of non-linear transformations, followed by a normal Perceptron-style layer. The scheme presented here has more in common with the work reviewed in Barron (88) (see also Tenorio 90) on polynomial networks in that it uses low order polynomials in a tree of low degree. The differences lie in a global rather than layer-by-layer learning scheme, and a transfer function derived from a gaussian discriminant function.

## 2. THE ARTIFICIAL DENDRITIC TREE (ADT)

The network architecture in Figure 1(a) is that of a binary tree which propagates real number values from its leaf nodes (or inputs) to its root node which is the output. In this simple formulation, the tree is construed as a binary classifier. The output node signals a number between 1 and 0 which represents the probability that the pattern presented to the tree was a member of the positive class of patterns or the negative class. Because the input patterns may have extremely high dimension and the tree is, at least initially, constrained to be binary, the depth of the tree may be significant, at least more than one might like to back-propagate through. A transfer function is associated with each 'hidden' node of the tree and the output node. This will hereafter be referred to as a *Z-function*, for the simple reason that it takes in two variables X and Y, and outputs Z. A cascade of Z-functions performs the computation of the tree and the learning procedure consists of changing these functions. The tree is referred to as an *Artificial Dendritic Tree* or ADT with the same degree of licence that one may talk of Artificial Neural Networks, or ANNs.

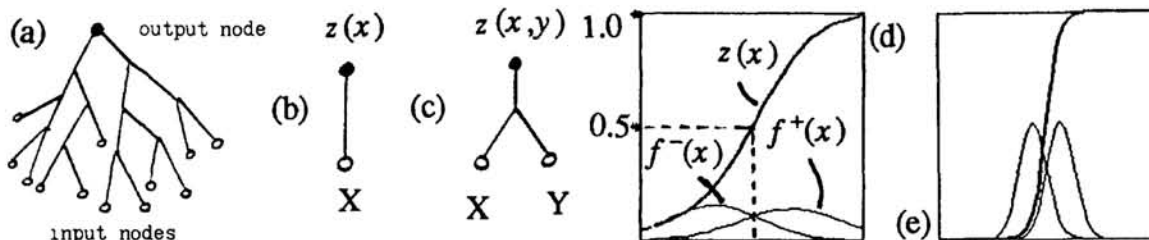

**Figure 1:** (a) an Artificial Dendritic Tree, (b) a 1D Z-node (c) a 2D Z-node (d) A 1D Z-function constructed from 2 gaussians (e) approximating a step function

## 2.1. THE TRANSFER FUNCTION

The idea behind the Z-function is to allow the two variables arriving at a node to interact locally in a non-linear way which contributes to the global computation of the tree. The transfer function is derived from statistical considerations. To simplify, consider the one-dimensional case of a variable X travelling on a wire as in Figure 1(b). A statistical estimation procedure could observe the distribution of values of X when the global pattern was positive or negative and derive a decision rule from these. In Figure 1(d), the two density functions $f^+(x)$ and $f^-(x)$ are plotted. Where they meet, the local computation must answer that, based on its information, the global pattern is positively classified with probability 0.5. Assuming that there are equal numbers of positive and negative patterns (ie: that the *a priori* probability of positive is 0.5), it is easy to see that the *conditional probability* of being in the positive class given our value for X, is given by equation (1).

$$z(x) = P[class=+ve \mid x] = \frac{f^+(x)}{f^+(x)+f^-(x)} \tag{1}$$

This can be also derived from Bayesian reasoning (Therrien, 89). The form of $z(x)$ is shown with the thick line in Figure 1(d) for the given $f^+(x)$ and $f^-(x)$. If $f^+(x)$ and $f^-(x)$ can be usefully approximated by normal (gaussian) curves as plotted above, then (1) translates into (2):

$$z(x) = \frac{1}{1+e^{-input}} \quad , \quad input = \beta^-(x) - \beta^+(x) + \ln\left[\frac{\alpha^-}{\alpha^+}\right] \tag{2}$$

This can be obtained by substituting equation (4) overleaf into (1) using the definitions of $\alpha$ and $\beta$ given. The exact form $\alpha$ and $\beta$ take depends on the number of variables input. The first striking thing is that the form of (2) is exactly that of the back-propagation logistic function. The second is that *input* is a polynomial quadratic expression. For Z-functions with 2 inputs $(x,y)$ using formulas (4.2) it takes the form:

$$w_1x^2+w_2y^2+w_3xy+w_4x+w_5y+w_6 \tag{3}$$

The $w$'s can be thought of as weights just as in backprop, defining a 6D space of transfer functions. However optimising the $w$'s directly through gradient descent may not be the best idea (though this is what Tenorio does), since for any error function $E$, $\partial E/\partial w_4 = x\,\partial E/\partial w_1 = y\,\partial E/\partial w_3$. That is, the axes of the optimisation are not independent of each other. There are, however, two sets of 5 independent parameters which the $w$'s in (3) are actually composed from if we calculate *input* from (4.2). These are $\mu_x^+$, $\sigma_x^+$, $\mu_y^+$, $\sigma_y^+$ and $r^+$, denoting the means, standard deviations and correlation coefficient defining the two-dimensional distribution of $(x,y)$ values which should be positively classified. The other 5 variables define the negative distribution.

Thus 2 Gaussians (hereafter referred to as the *positive* and *negative models*) define a quadratic transfer function (called the *Z-function*) which can be interpreted as expressing conditional probability of positive class membership. The shape of these functions can be altered by changing the statistical parameters defining the distributions which underly them. In Figure 1(d), a 1-dimensional Z-function is seen to be sigmoidal though it need not be monotonic at all. Figure 2(b)-(h) shows a selection of 2D Z-functions. In general the Z-function divides its N-dimensional input space with a N-1 dimensional hypersurface. In 2D, this will be an ellipse, a parabola, a hyperbola or some combination of the three. Although the dividing surface is quadratic, the Z-function is still a logistic or squashing function. The exponent *input* is actually equivalent to the *log likelihood ratio* or $\ln(f^+(x)/f^-(x))$, commonly used in statistics.

In this work, 2-dimensional gaussians are used to generate Z-functions. There are compelling reasons for this. One dimensional Z-functions are of little use since they do not reduce information. Z-functions of dimension higher than 1 perform optimal class-based information reduction by propagating conditional probabilities of class membership. But 2D Z-functions using 2D gaussians are of particular interest because they include in their function space all boolean functions of two variables (or at least analogue versions of these functions). For example the gaussians which would come to represent the positive and negative exemplar patterns for XOR are drawn as ellipses in Figure 2(a). They have equal means and variances but the negative exemplar patterns are correlated while the positive ones are anti-correlated. These models automatically give rise to the XOR surface in Figure 2(b) if put through equation (2). An interesting

observation is that a problem of Nth order (XOR is 2nd order, 3-parity is 3rd order etc) can be solved by a polynomial of degree N (Figure 2d). Since 2nd degree polynomials like (3) are used in our system, there is one step up in power from 1st degree systems like the Perceptron. Thus 3-parity is to the Z-function unit what XOR is to the Perceptron (in this case not *quadratically separable*).

$$A\ GAUSSIAN\ IS: \qquad f(x)=\frac{1}{\alpha}e^{-\beta(x)} \tag{4}$$

$$in\ one\ dimension: \qquad \alpha=(2\pi)^{1/2}\sigma_x \tag{4.1.1}$$

$$\beta(x)=\frac{(x-\mu_x)^2}{2\sigma_x^2} \tag{4.1.2}$$

$$in\ two\ dimensions: \qquad \alpha=2\pi\sigma_x\sigma_y(1-r^2)^{1/2} \tag{4.2.1}$$

$$\beta(x,y)=\frac{1}{2(1-r^2)}\left[\frac{(x-\mu_x)^2}{\sigma_x^2}+\frac{(y-\mu_y)^2}{\sigma_y^2}-2r\frac{(x-\mu_x)(y-\mu_y)}{\sigma_x\sigma_y}\right] \tag{4.2.2}$$

$$in\ n\ dimensions: \qquad \alpha=(2\pi)^{n/2}|K|^{1/2} \tag{4.n.1}$$

$$\beta(\underline{x})=\frac{1}{2}(\underline{x}-\underline{m})^T K^{-1}(\underline{x}-\underline{m}) \tag{4.n.2}$$

$\mu_x=E[x]$ $\qquad$ *is the expected value or mean of x*

$\sigma_x^2=E[x^2]-\mu_x^2$ $\qquad$ *is the variance of x*

$r=\dfrac{E[xy]-\mu_x\mu_y}{\sigma_x\sigma_y}$ $\qquad$ *is the correlation coefficient of a bivariate gaussian*

$\underline{m}=E[\underline{x}]$ $\qquad$ *is the mean vector of a multivariate gaussian*

$K=E[(\underline{x}-\underline{m})(\underline{x}-\underline{m})^T]$ $\qquad$ *is the covariance matrix of a multivariate gaussian with* $|K|$ *its determinant*

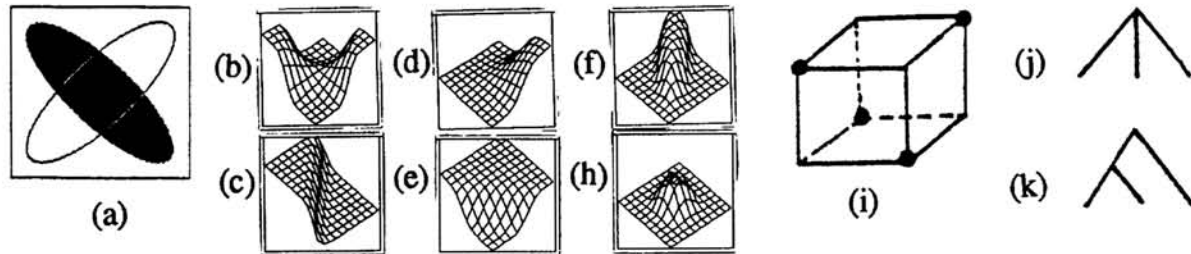

**Figure 2:** (a) two anti-correlated gaussians seen from above (b) the resulting Z-function (c)-(h) Some other 2D Z-functions. (i) 3-parity in a cube cannot be solved by a 3D Z-function (j) but yields to a cascade of 2D ones (k).

## 2.2. THE LEARNING PROCEDURE

If gaussians are used to model the distribution of inputs $x$ which give positive and negative classification *errors*, rather than just the distribution of positively and negatively classified $x$, then it is possible to formulate an incremental learning procedure for training Z-functions. This procedure enables the system to deal with data which is not gaussianly distributed.

### 2.2.1.  Without hidden units: learning a step function.

A simple example illustrates this principle. Consider a network consisting entirely of a 1-dimensional Z-function, as in Figure 1(b). The input is a real number from 0 to 1 and the output is to be a step function, such that 0.5-1.0 is classed positively (output 1.0) and 0.0-0.5 should output 0.0. The 4 parameters of the Z-function ($\mu^+,\mu^-,\sigma^+,\sigma^-$) are initialised randomly and example patterns are presented to the 'tree'. On each presentation $t$, the error $\delta$ in the response is calculated by $\delta_t \leftarrow d_t - o_t$, the desired minus the actual output at time $t$, and 2 of the parameters are altered. If the error is positive, the positive model is altered, otherwise the negative model is altered. Changing a model consists of 'sliding' the estimates of the appropriate first and second moments ($E[x]$ and $E[x^2]$) according to a 'moving-average' scheme:

$$E[x]_t \leftarrow \varepsilon\delta_t x_t + (1-\varepsilon\delta_t)E[x]_{t-1} \tag{5.1}$$

$$E[x^2]_t \leftarrow \varepsilon\delta_t x_t^2 + (1-\varepsilon\delta_t)E[x^2]_{t-1} \tag{5.2}$$

where $\varepsilon$ is a plasticity or learning rate, $x_t$ is the value input and $E[x]_{t-1}$ was the previous estimate of the mean value of $x$ for the appropriate gaussian. This rule means that at any moment, the parameters determining the positive and negative models are weighted averages of recent inputs which have generated errors. The influence which a particular input has had decays over time. This algorithm was run with $\varepsilon=0.1$. After 100 random numbers had been presented, with error signals from the step-function changing the models, the models come to well represent the distribution of positive and negative inputs. At this stage the models and their associated Z-function are those shown in Figure 1(d). But now, most of the error reinforcement will be coming from a small region around 0.5, which means that since the gaussians are modelling the errors, they will be drawn towards the centre and become narrower. This has the effect, Figure 1(e), of increasing the gain of the sigmoidal Z-function. In the limit, it will converge to a perfect step function as the gaussians become infinitesimally separated delta functions. This initial demonstration shows the *automatic gain adjustment* property of the Z-function.

### 2.2.2.  With hidden units: the 6-multiplexer.

The first example showed how a 1D Z-function can minimise error by modelling it. This example shows how a cascade of 2-dimensional Z-functions can co-operate to solve a 3rd order problem. A 6-multiplexer circuit receives as input 6 bits, 4 of which are data bits and 2 are address bits. If the address bits are 00, it must output the contents of the first data bit, if 01, the second, 10 the third and 11 the fourth. There are 64 different input patterns. Choosing an tree architecture is a difficult problem in general, but the first step is to choose one which we know can solve the problem. This is illustrated in Figure 3(a). This is an architecture for which there exists a solution using binary Boolean functions.

The tree's solution was arrived at as follows: each node was initialised with 10 random values: $E[x]$, $E[y]$, $E[x^2]$, $E[y^2]$ and $E[xy]$ for each of its positive and negative models. The learning rate $\varepsilon$ was set to 0.02 and input patterns were generated and propagated up to the top node, where an error measurement was made. The error was then broadcast *globally to all nodes*, each one, in effect, being told to respond more positively (or negatively) should the same circumstances arise again, and adjusting their Z-functions in the same way as equations (5). This time, however, 5 parameters

were adjusted per node per presentation, instead of 2. Again, which model (positive or negative) is adjusted depends on the sign of the error at the top of the tree.

The tree learns after about 200 random bit patterns are presented (7 seconds on a Symbolics). After 300 presentations (the state depicted in Figure 3a), the mean squared error is falling steadily to zero. An adequate back-propagation network takes 6000 presentations to converge on a solution. The solution achieved is a rather messy combination of half-hearted XORs and NXORs, and ambiguous AND/ORs. The problem was tried with different trees. In general any tree of sufficient richness can solve the problem though larger trees take longer. Trees for which no nice solutions exist, ie: those with fewer than 6 well-chosen inputs from the address bits can sometimes still perform rather well. A tree with straight convergence, only one contact per address bit, can still quickly approach 80% performance, but further training is destructive. Figure 3(b) shows a tree trained to output 1 if half or more of its 8 inputs were on.

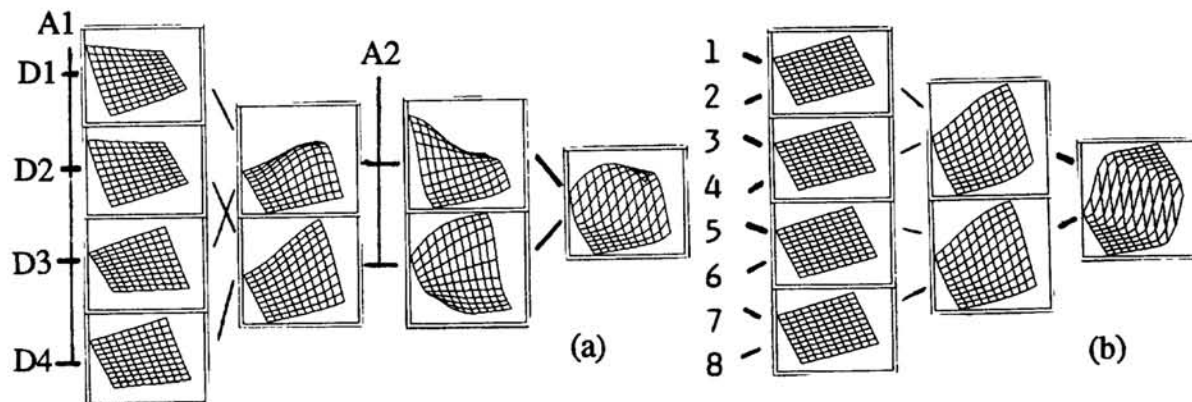

**Figure 3:** Solving the 6-multiplexer (a) and the 8-majority predicate (b)

### 2.2.3. Phoneme classification.

A good question was if such a tree could perform well on a large problem, so a typical back-propagation application was attempted. Space does not permit a full account here, but the details appear in Bell (89). The data came from 100 speakers speaking the confusable E-set phonemes (B, D, E and V). This was the same data as that used by Lang & Hinton (88). Four trees were built out of 192 input units and the trees trained using a learning schedule of $\varepsilon$ falling from 0.01 to 0.001 over the course of 30 presentations of each of 668 training patterns. Generalisation to a test set was 88.5%, 0.5% worse than an equivalently simple backprop net. A more sophisticated backprop net, using time-delays and multiresolution training could reach 93% generalisation. Thirty epochs with the trees took some 16 hours on a Sun 3-260 whereas the backprop experiments were performed on a Convex supercomputer. The conclusion from these experiments is that trees some 8 levels deep are capable of almost matching normal back-propagation on a large classification task in a fraction of the training time. Attempts to build time-symmetry into the trees have not so far been successful.

## 3. DISCUSSION

Even within the context of other connectionist learning procedures, there is something of an air of mystery about this one. The apparatus of gradient descent, either for individual units or for the whole tree is absent or at least hidden.

## 3.1. HOW DOES IT WORK?

It is necessary to reflect on the effect of modelling errors. Models of errors are an attempt to push a node's outputs towards the edge of its parent's input square. Where the model is perfect, it is simple for the node above to model the model by applying a sigmoid, and so on to the top of the tree, where the error disappears. But the modelling is actually done in a totally distributed and collaborative way. The identification of 1.0 with positive error (top output too small) means that Z-functions are more likely to be monotonic towards (1,1) the further they are from the inputs.

Two standard problems are overcome in unusual ways. The first, *credit assignment*, is solved because different Z-functions are able to model different errors, giving them different roles. Although all nodes receive the same feedback, some changes to a node's model will be swiftly undone when the new errors that result from them begin to be broadcast. Other nodes can change freely either because they are not yet essential to the computation or because there exist alterations of their models tolerable to the nodes above. The second problem is *stability*. In backprop, the way the error diffuses through the net ensures that the upper weights are slaved to the lower ones because the lower are changing more slowly. In this system, the upper nodes are slaved to the lower ones  because they are explicitly modelling their activities. Conversely, the lower nodes will never be allowed to change too quickly since the errors generated by sluggish top nodes will throw them back into the behaviour the top nodes expect. For a low enough learning rate $\varepsilon$, the solutions are stable.

Amongst the real problems with this system are the following. First, the credit assignment is not solved for units receiving the same input variables, making many normal connectionist architectures impossible. Second, the system can only deal with 2 classes. Third, as with other algorithms, choice of architecture is a 'black art'.

## 3.2. BIOPHYSICS & REAL NEURONS

The name 'Artificial Dendritic Tree' is perhaps overdoing it. The tree has no dynamic properties, activation flows in only one direction, the branchpoints of the tree routinely implement XOR and the 'cell' as a whole implements phoneme recognition (only a small step from grandmothers). The title was kept because what drove the work was a search for a computational explanation of how fine-grained local non-linearities of low degree could combine in a learning process. Work in computational neuroscience, in particular with compartmental models (Koch & Poggio 87; Rall & Segev 88; Segev et al 89, Shepherd & Brayton 87) have shown that it is likely that many non-linear effects take place between synapse and soma. Synaptic transfer functions can be sigmoidal, spines with active channels may mutually excite each other (even implement boolean computations) and inhibitory inputs can 'veto' firing in a highly non-linear fashion (silent inhibition). The dendritic membrane itself is filled with active ion channels, whose boosting or quenching properties depend in a complex way on the intracellular voltage levels or $Ca^{2+}$ concentration (itself dependent on voltage). Thus we may be able to consider the membrane itself as a distributed processing system, meaning that the synapses are no longer the privileged sites of learning which they have tended to be since Hebb. Active channels can serve to implement threshold functions just as well at the dendritic branchpoints as at the soma, where they generate spikes. There are many different kinds of ion channel (Yamada et al, 89) with inhomogenous distributions over the dendritic tree. A neuron's DNA may generate a certain 'base set' of channel proteins that span a non-linear function space just as our

parameters span the Z-function space. The properties of a part of dendritic membrane could be seen as a point in *channel space*. Viewed this way, the neuron becomes one large computer. When one considers the Purkinje cell of the cerebellum with 100,000 inputs, as many spines, a massive arborisation full of active channels, many of them Ca-permeable or Ca-dependent, with spiking and plateau potentials occurring in the dendritic tree, the notion that the cell may be implementing a 99,999 dimensional hyperplane starts to recede. here is an extra motivation for considering the cell as a complex computer. Algorithms such as back-propagation would require feedback circuits to send error. If the cell is the feedback unit, then reinforcement can occur as a spike at the soma reinvades the dendritic tree. Thus nerves may not spike just for axonal purposes, but also to penetrate the electrotonic length of the dendrites. This was thought to be a component of Hebbian learning at the synapses, but it could be the basis of more if the dendritic membrane computes.

## 4. Acknowledgements

To Kevin Lang for the speech data and to Rolf Pfeifer and Luc Steels for support. Further credits in Bell (90). The author is funded by ESPRIT B.R.A. 3234.

## 5. References

Barron A & Barron R (88) Statistical Learning Networks: a unifying view, in Wegman E (ed) *Proc. 20th Symp. on Comp. Science & Statistics* [see also this volume]

Bell T (89) Artificial Dendritic Learning, in Almeida L. (ed) *Proc. EURASIP Workshop on Neural Networks*, Lecture notes in Computer Science, Springer-Verlag. [also VUB AI-lab Memo 89-20].

Durbin R & Rumelhart D (89) Product Units: A Computationally Powerful and Biologically Plausible Extension to Backpropagation Nets, *Neural Computation 1*

Giles C.L. & Maxwell T (87) Learning, invariance and generalisation in high-order neural networks, *Applied Optics* vol 26, no. 23

Koch C & Poggio T (87) Biophysics of Computational Systems: Neurons, synapses and membranes, in G. Edelman et al (eds), *Synaptic Function*, John Wiley.

Lang K & Hinton G (88) The Development of the Time-Delay Neural Network Architecture for Speech Recognition, *Tech Report CMU-CS-88-152*

Rall W & Segev I (88) Excitable Dendritic Spine Clusters: non-linear synaptic processing, in R.Cotterill (ed) *Computer Simulation in Brain Science*, Camb.U.P.

Segev I, Fleshman J & Burke R. (89) Compartmental Models of Complex Neurons, in *Methods in Neuronal Modelling*

Shepherd G & Brayton R (87) Logic operations are properties of computer simulated interactions between excitable dendritic spines, *Neuroscience*, vol 21, no. 1 1987 Koch C & Segev I (eds) MIT press 1989

Tenorio M & Lee W (90) Self-Organizing Network for Optimal Supervised Learning, *IEEE Transactions in Neural Networks*, 1990 [see also this volume]

Therrien C (89) *Decision Estimation and Classification.*

Yamada W, Koch C & Adams P (89) Multiple Channels and Calcium Dynamics, in *Methods in Neuronal Modelling* Koch C & Segev I (eds) MIT press 1989.